# 3D Object Detection and Viewpoint Estimation with a Deformable 3D Cuboid Model

**Sanja Fidler**
TTI Chicago
fidler@ttic.edu

**Sven Dickinson**
University of Toronto
sven@cs.toronto.edu

**Raquel Urtasun**
TTI Chicago
rurtasun@ttic.edu

## Abstract

This paper addresses the problem of category-level 3D object detection. Given a monocular image, our aim is to localize the objects in 3D by enclosing them with tight oriented 3D bounding boxes. We propose a novel approach that extends the well-acclaimed deformable part-based model [1] to reason in 3D. Our model represents an object class as a deformable 3D cuboid composed of faces and parts, which are both allowed to deform with respect to their anchors on the 3D box. We model the appearance of each face in fronto-parallel coordinates, thus effectively factoring out the appearance variation induced by viewpoint. Our model reasons about face visibility patters called aspects. We train the cuboid model jointly and discriminatively and share weights across all aspects to attain efficiency. Inference then entails sliding and rotating the box in 3D and scoring object hypotheses. While for inference we discretize the search space, the variables are continuous in our model. We demonstrate the effectiveness of our approach in indoor and outdoor scenarios, and show that our approach significantly outperforms the state-of-the-art in both 2D [1] and 3D object detection [2].

## 1   Introduction

Estimating semantic 3D information from monocular images is an important task in applications such as autonomous driving and personal robotics. Let's consider for example, the case of an autonomous agent driving around a city. In order to properly react to dynamic situations, such an agent needs to reason about which objects are present in the scene, as well as their 3D location, orientation and 3D extent. Likewise, a home robot requires accurate 3D information in order to navigate in cluttered environments as well as grasp and manipulate objects.

While impressive performance has been achieved for instance-level 3D object recognition [3], category-level 3D object detection has proven to be a much harder task, due to intra-class variation as well as appearance variation due to viewpoint changes. The most common approach to 3D detection is to discretize the viewing sphere into bins and train a 2D detector for each viewpoint [4, 5, 1, 6]. However, these approaches output rather weak 3D information, where typically a 2D bounding box around the object is returned along with an estimated discretized viewpoint.

In contrast, object-centered approaches represent and reason about objects using more sophisticated 3D models. The main idea is to index (or vote) into a parameterized pose space with local geometric [7] or appearance features, that bear only weak viewpoint dependencies [8, 9, 10, 11]. The main advantage of this line of work is that it enables a continuous pose representation [10, 11, 12, 8], 3D bounding box prediction [8], and potentially requires less training examples due to its more com-

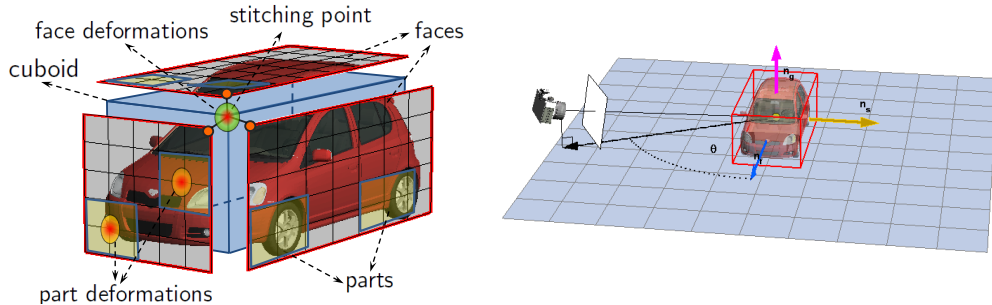

Figure 1: **Left:** Our deformable 3D cuboid model. **Right** Viewpoint angle $\theta$.

pact visual representation. Unfortunately, these approaches work with weaker appearance models that cannot compete with current discriminative approaches [1, 6, 13]. Recently, Hedau et al. [2] proposed to extend the 2D HOG-based template detector of [14] to predict 3D cuboids. However, since the model represents object's appearance as a rigid template in 3D, its performance has been shown to be inferior to (2D) deformable part-based models (DPMs) [1].

In contrast, in this paper we extend DPM to reason in 3D. Our model represents an object class with a deformable 3D cuboid composed of faces and parts, which are both allowed to deform with respect to their anchors on the 3D box (see Fig 1). Towards this goal, we introduce the notion of *stitching point*, which enables the deformation between the faces and the cuboid to be encoded efficiently. We model the appearance of each face in fronto-parallel coordinates, thus effectively factoring out the appearance variation due to viewpoint. We reason about different face visibility patterns called *aspects* [15]. We train the cuboid model jointly and discriminatively and share weights across all aspects to attain efficiency. In inference, our model outputs 2D along with oriented 3D bounding boxes around the objects. This enables the estimation of object's viewpoint which is a continuous variable in our representation. We demonstrate the effectiveness of our approach in indoor [2] and outdoor scenarios [16], and show that our approach significantly outperforms the state-of-the-art in both 2D [1] and 3D object detection [2].

## 2 Related work

The most common way to tackle 3D detection is to represent a 3D object by a collection of independent 2D appearance models [4, 5, 1, 6, 13], one for each viewpoint. Several authors augmented the multi-view representation with weak 3D information by linking the features or parts across views [17, 18, 19, 20, 21]. This allows for a dense representation of the viewing sphere by morphing related near-by views [12]. Since these methods usually require a significant amount of training data, renderings of synthetic CAD models have been used to supplement under-represented views or provide supervision for training object parts or object geometry [22, 13, 8].

Object-centered approaches, represent object classes with a 3D model typically equipped with view-invariant geometry and appearance [7, 23, 24, 8, 9, 10, 11, 25]. While these types of models are attractive as they enable continuous viewpoint representations, their detection performance has typically been inferior to 2D deformable models.

Deformable part-based models (DPMs) [1] are nowadays arguably the most successful approach to category-level 2D detection. Towards 3D, DPMs have been extended to reason about object viewpoint by training the mixture model with viewpoint supervision [6, 13]. Pepik et al. [13] took a step further by incorporating supervision also at the part level. Consistency was enforced by forcing the parts for different 2D viewpoint models to belong to the same set of 3D parts in the physical space. However, all these approaches base their representation in 2D and thus output only 2D bounding boxes along with a discretized viewpoint.

The closest work to ours is [2], which models an object with a rigid 3D cuboid, composed of independently trained faces without deformations or parts. Our model shares certain similarities with this work, but has a set of important differences. First, our model is hierarchical and deformable: we allow deformations of the faces, while the faces themselves are composed of deformable parts. We also explicitly reason about the visibility patterns of the cuboid model and train the model accordingly. Furthermore, all the parameters in our model are trained jointly using a latent SVM formulation. These differences are important, as our approach outperforms [2] by a significant margin.

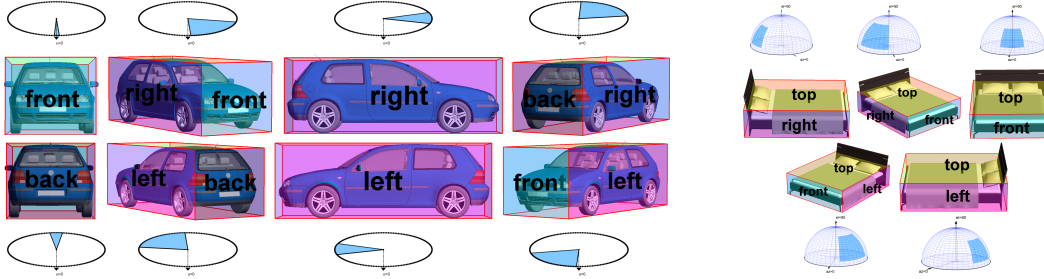

Figure 2: Aspects, together with the range of $\theta$ that they cover, for (**left**) cars and (**right**) beds.

Finally, in concurrent work, Xiang and Savarese [26] introduced a deformable 3D aspect model, where an object is represented as a set of planar parts in 3D. This model shares many similarities with our approach, however, unlike ours, it requires a collection of CAD models in training.

## 3   A Deformable 3D Cuboid Model

In this paper, we are interested in the problem of, given a single image, estimating the 3D location and orientation of the objects present in the scene. We parameterize the problem as the one of estimating a tight 3D bounding box around each object. Our 3D box is oriented, as we reason about the correspondences between the faces in the estimated bounding box and the faces of our model (i.e., which face is the top face, front face, etc). Towards this goal, we represent an object class as a deformable 3D cuboid, which is composed of 6 deformable faces, i.e., their locations and scales can deviate from their anchors on the cuboid. The model for each cuboid's face is a 2D template that represents the appearance of the object in view-rectified coordinates, i.e., where the face is frontal. Additionally, we augment each face with parts, and employ a deformation model between the locations of the parts and the anchor points on the face they belong to. We assume that any viewpoint of an object in the image domain can be modeled by rotating our cuboid in 3D, followed by perspective projection onto the image plane. Thus inference involves sliding and rotating the deformable cuboid in 3D and scoring the hypotheses.

A necessary component of any 3D model is to properly reason about the face visibility of the object (in our case, the cuboid). Assuming a perspective camera, for any given viewpoint, at most 3 faces are visible in an image. Topologically different visibility patterns define different *aspects* [15] of the object. Note that a cuboid can have up to 26 aspects, however, not all necessarily occur for each object class. For example, for objects supported by the floor, the bottom face will never be visible. For cars, typically the top face is not visible either. Our model only reasons about the occurring aspects of the object class of interest, which we estimate from the training data. Note that the visibility, and thus the aspect, is a function of the 3D orientation and position of a cuboid hypothesis with respect to the camera. We define $\theta$ to be the angle between the outer normal to the front face of the cuboid hypothesis, and the vector connecting the camera and the center of the 3D box. We refer the reader to Fig. 1 for a visualization. Assuming a camera overlooking the center of the cuboid, Fig. 2 shows the range of the cuboid orientation angle on the viewing sphere for which each aspect occurs in the datasets of [2, 16], which we employ for our experiments. Note however, that in inference we do not assume that the object's center lies on the camera's principal axis.

In order to make the cuboid deformable, we introduce the notion of *stitching point*, which is a point on the box that is common to all visible faces for a particular aspect. We incorporate a quadratic deformation cost between the locations of the faces and the stitching point to encourage the cuboid to be as rigid as possible. We impose an additional deformation cost between the visible faces, ensuring that their sizes match when we stitch them into a cuboid hypothesis. Our model represents each aspect with its own set of weights. To reduce the computational complexity and impose regularization, we share the face and part templates across all aspects, as well as the deformations between them. However, the deformations between the faces and the cuboid are aspect specific as they depend on the stitching point.

We formally define the model by a $(6 \cdot (n+1)+1)$-tuple $(\{(P_i, P_{i,1}, \ldots, P_{i,n})\}_{i=1,\ldots,6}, b)$ where $P_i$ models the $i$-th face, $P_{i,j}$ is a model for the $j$-th part belonging to face $i$, and $b$ is a real valued bias term. For ease of exposition, we assume each face to have the same number of parts, $n$; however, the framework is general and allows the numbers of parts to vary across faces. For each aspect $a$,

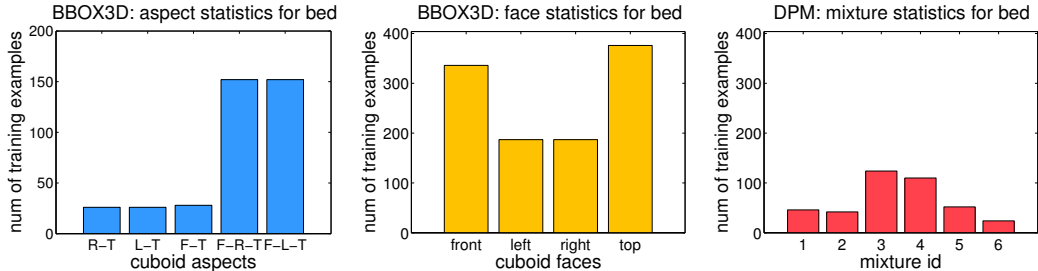

Figure 3: Dataset [2] statistics for training our cuboid model (left and middle) and DPM [1] (right).

we define each of its visible faces by a 3-tuple $(F_i, r_{a,i}, d_{a,i}^{stitch}, b_a)$, where $F_i$ is a filter for the $i$-th face, $r_{a,i}$ is a two-dimensional vector specifying the position of the $i$-th face relative to the position of the stitching point in the rectified view, and $d_i$ is a four-dimensional vector specifying coefficients of a quadratic function defining a deformation cost for each possible placement of the face relative to the position of the stitching point. Here, $b_a$ is a bias term that is aspect specific and allows us to calibrate the scores across aspects with different number of visible faces. Note that $F_i$ will be shared across aspects and thus we omit index $a$.

The model representing each part is face-specific, and is defined by a 3-tuple $(F_{i,j}, r_{i,j}, d_{i,j})$, where $F_{i,j}$ is a filter for the $j$-th part of the $i$-th face, $r_{i,j}$ is a two-dimensional vector specifying an "anchor" position for part $j$ relative to the root position of face $i$, and $d_{i,j}$ is a four dimensional vector specifying coefficients of a quadratic function defining a deformation cost for each possible placement of the part relative to the anchor position on the face. Note that the parts are defined relative to the face and are thus independent of the aspects. We thus share them across aspects.

The appearance templates as well as the deformation parameters in the model are defined for each face in a canonical view where that face is frontal. We thus score a face hypothesis in the rectified view that makes the hypothesis frontal. Each pair of parallel faces shares a homography, and thus at most three rectifications are needed for each viewpoint hypothesis $\theta$. In *indoor scenarios*, we estimate the 3 orthogonal vanishing points and assume a Manhattan world. As a consequence only 3 rectifications are necessary altogether. In the *outdoor scenario*, we assume that at least the vertical vanishing point is given, or equivalently, that the orientation (but not position) of the ground plane is known. As a consequence, we only need to search for a 1-D angle $\theta$, i.e., the azimuth, in order to estimate the rotation of the 3D box. A sliding window approach is then used to score the cuboid hypotheses, by scoring the parts, faces and their deformations in their own rectified view, as well as the deformations of the faces with respect to the stitching point.

Following 2D deformable part-based models [1], we use a pyramid of HOG features to describe each face-specific rectified view, $H(i, \theta)$, and score a template for a face as follows:

$$score(p_i, \theta) = \sum_{u', v'} F_i(u', v') \cdot H[u_i + u'; v_i + v'; i, \theta] \qquad (1)$$

where $p_i = (u_i, v_i, l_i)$ specifies the position $(u_i, v_i)$ and level $l_i$ of the face filters in the face-rectified feature pyramids. We score each part $p_{i,j} = (u_{i,j}, v_{i,j}, l_{i,j})$ in a similar fashion, but the pyramid is indexed at twice the resolution of the face. We define the compatibility score between the parts and the corresponding face, denoted as $\mathbf{p}_i = \{p_i, \{p_{i,j}\}_{j=1,...,n}\}$, as the sum over the part scores and their deformations with respect to the anchor positions on the face:

$$score_{parts}(\mathbf{p}_i, \theta) = \sum_{j=1}^{n} \left( score(p_{i,j}, \theta) - d_{ij} \cdot \phi_d(p_i, p_{i,j}) \right), \qquad (2)$$

We thus define the score of a 3D cuboid hypothesis to be the sum of the scores of each face and its parts, as well as the deformation of each face with respect to the stitching point and the deformation of the faces with respect to each other as follows

$$score(x, \theta, \mathbf{s}, \mathbf{p}) = \sum_{i=1}^{6} V(i, a) \left( score(p_i, \theta) - d_{a,i}^{stitch} \cdot \phi_d^{stich}(p_i, \mathbf{s}, \theta) \right) -$$

$$- \sum_{i>ref}^{6} V(i, a) \cdot d_{i,ref}^{face} \phi_d^{face}(p_i, p_{ref}, \theta) + \sum_{i=1}^{6} V(i, a) \cdot score_{parts}(\mathbf{p}_i, \theta) + b_a$$

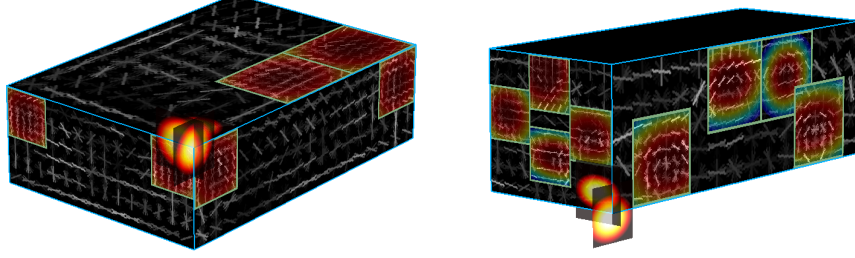

Figure 4: Learned models for (**left**) bed, (**right**) car.

where $\mathbf{p} = (\mathbf{p}_1, \cdots, \mathbf{p}_6)$ and $V(i, a)$ is a binary variable encoding whether face $i$ is visible under aspect $a$. Note that $a = a(\theta, \mathbf{s})$ can be deterministically computed from the rotation angle $\theta$ and the position of the stitching point $\mathbf{s}$ (which we assume to always be visible), which in turns determines the face visibility $V$. We use $ref$ to index the first visible face in the aspect model, and

$$\phi_d(p_i, p_{i,j}, \theta) = \phi_d(du, dv) = (du, dv, du^2, dv^2) \tag{3}$$

are the part deformation features, computed in the rectified image of face $i$ implied by the 3D angle $\theta$. As in [1], we employ a quadratic deformation cost to model the relationships between the parts and the anchor points on the face, and define $(du_{i,j}, dv_{i,j}) = (u_{i,j}, v_{i,j}) - (2 \cdot (u_i, v_i) + r_{i,j})$ as the displacement of the $j$-th part with respect to its anchor $(u_i, v_i)$ in the rectified $j$-th face. The deformation features $\phi_d^{stich}(p_i, \mathbf{s}, \theta)$ between the face $p_i$ and the stitching point $\mathbf{s}$ are defined as $(du_i, dv_i) = (u_i, v_i) - (u(\mathbf{s}, i), v(\mathbf{s}, i)) + r_{a,i})$. Here, $(u(\mathbf{s}, i), v(\mathbf{s}, i))$ is the position of the stitching point in the rectified coordinates corresponding to face $i$ and level $l$.

We define the deformation cost between the faces to be a function of their relative dimensions:

$$\phi_d^{face}(p_i, p_k, \theta) = \begin{cases} 0, & \text{if } \frac{max(e_i, e_k)}{min(e_i, e_k)} < 1 + \epsilon \\ \infty & \text{otherwise} \end{cases} \tag{4}$$

with $e_i$ and $e_k$ the lengths of the common edge between faces $i$ and $k$. We define the deformation of a face with respect to the stitching point to also be quadratic. It is defined in the rectified view, and thus depend on $\theta$. We additionally incorporate a bias term for each aspect, $b_a$, to make the scores of multiple aspects comparable when we combine them into a full cuboid model.

Given an image $x$, the score of a hypothesized 3D cuboid can be obtained as the dot product between the model's parameters and a feature vector, i.e., $score(x, \theta, \mathbf{s}, \mathbf{p}) = \mathbf{w}_a \cdot \Phi(x, a(\theta, \mathbf{s}), \mathbf{p})$, with

$$\mathbf{w}_a = (F'_1, \cdots, F'_6, F'_{1,1}, \cdots, F'_{6,n}, d_{1,1}, \cdots, d_{6,n}, d_{a,1}^{stitch}, \cdots, d_{a,6}^{stitch}, d_{1,2}^{face}, \cdots, d_{5,6}^{face}, b_a), \tag{5}$$

and the feature vector:

$$\Phi(x, a(\theta, \mathbf{s}), \mathbf{p}) = \big( \hat{H}(p_1, i, \theta), \cdots, \hat{H}(p_{1,1}, i, \theta), -\hat{\phi}_d(p_1, p_{1,1}), \cdots, -\hat{\phi}_d(p_6, p_{6,n}),$$
$$- \hat{\phi}_d^{stitch}(p_1, \mathbf{s}, \theta), \cdots, -\hat{\phi}_d^{stitch}(p_6, \mathbf{s}, \theta), -\hat{\phi}_d^{face}(p_1, p_2), \cdots, 1 \big)$$

where $\hat{\phi}$ includes the visibility score in the feature vector, e.g., $\hat{\phi}(i, \cdot) = V(i, a) \cdot \phi(i, \cdot)$.

**Inference:**    Inference in this model can be done by computing

$$f_w(x) = \max_{\theta, \mathbf{s}, \mathbf{p}} \mathbf{w}_a \cdot \Phi(x, a(\theta, \mathbf{s}), \mathbf{p})$$

This can be solved exactly via dynamic programming, where the score is first computed for each $\theta$, i.e., $\max_{\mathbf{s},\mathbf{p}} \mathbf{w}_a \cdot \Phi(x, a(\theta, \mathbf{s}), \mathbf{p})$, and then a max is taken over the angles $\theta$. We use a discretization of $20 \deg$ for the angles. To get the score for each $\theta$, we first compute the feature responses for the part and face templates (Eq. (1)) using a sliding window approach in the corresponding feature pyramids. As in [1], distance transforms are used to compute the deformation scores of the parts efficiently, that is, Eq. (2). The score for each face simply sums the response of the face template and the scores of the parts. We again use distance transforms to compute the deformation scores for each face and the stitching point, which is carried out in the rectified coordinates for each face. We then compute the deformation scores between the faces in Eq. (4), which can be performed efficiently due to the fact that sides of the same length along one dimension (horizontal or vertical) in the coordinates of face $i$ will also be constant along the corresponding line when projected to the coordinate system of face $j$. Thus, computing the side length ratios of two faces is not quadratic in the number of pixels but only in the number of horizontal or vertical lines. Finally, we reproject the scores to the image coordinate system and sum them to get the score for each $\theta$.

| | Detectors' performance | | | Layout rescoring | | |
|---|---|---|---|---|---|---|
| | DPM [1] | 3D det. | combined | DPM [1] | 3D det. | combined |
| Hedau et al. [2] | 54.2% | 51.3% | 59.6% | - | - | 62.8% |
| ours | 55.6% | 59.4% | 60.5% | 60.0% | 64.6% | 63.8% |

Table 1: Detection performance (measured in AP at 0.5 IOU overlap) for the bed dataset of [2]

| 3D measure | DPM fit3D | BBOX3D | combined | BBOX3D + layout | comb. + layout |
|---|---|---|---|---|---|
| **convex hull** | 48.2% | 53.9% | 53.9% | 57.8% | 57.1% |
| **face overlap** | 16.3% | 33.0% | 34.4% | 33.5% | 33.6% |

Table 2: 3D detection performance in AP (50% IOU overlap of convex hulls and faces)

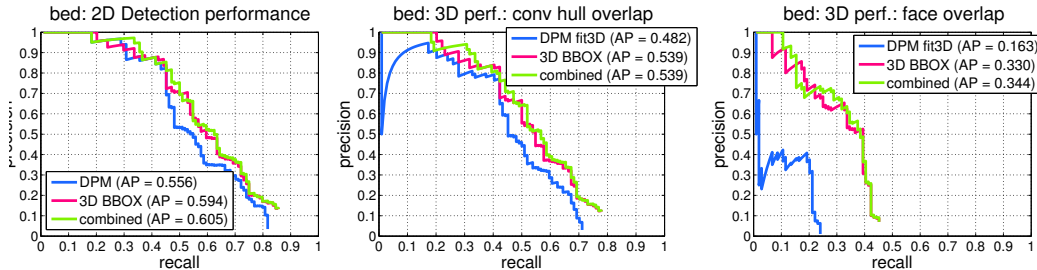

Figure 5: Precision-recall curves for (**left**) 2D detection (**middle**) convex hull, (right) face overlap.

**Learning:** Given a set of training samples $D = (\langle x_1, y_1, bb_1 \rangle, \cdots \langle x_N, y_N, bb_N \rangle)$, where $x$ is an image, $y_i \in \{-1, 1\}$, and $bb \in \mathbb{R}^{8 \times 2}$ are the eight coordinates of the 3D bounding box in the image, our goal is to learn the weights $\mathbf{w} = [\mathbf{w}_{a_1}, \cdots, \mathbf{w}_{a_P}]$ for all $P$ aspects in Eq. (5). To train our model using partially labeled data, we use a latent SVM formulation [1], however, frameworks such as latent structural SVMs [27] are also possible. To initialize the full model, we first learn a deformable face+parts model for each face independently, where the faces of the training examples are rectified to be frontal prior to training. We estimate the different aspects of our 3D model from the statistics of the training data, and compute for each training cuboid the relative positions $v_{a,i}$ of face $i$ and the stitching point in the rectified view of each face. We then perform joint training of the full model, treating the training cuboid and the stitching point as latent, however, requiring that each face filter and the face annotation overlap more than $70\%$. Following [1], we utilize a stochastic gradient descent approach which alternates between solving for the latent variables and updating the weights $\mathbf{w}$. Note that this algorithm is only guaranteed to converge to a local optimum, as the latent variables make the problem non-convex.

## 4   Experiments

We evaluate our approach on two datasets, the dataset of [2] as well as KITTI [16], an autonomous driving dataset. To our knowledge, these are the only datasets which have been labeled with 3D bounding boxes. We begin our experimentation with the indoor scenario [2]. The bedroom dataset contains 181 train and 128 test images. To enable a comparison with the DPM detector [1], we trained a model with 6 mixtures and 8 parts using the same training instances but employing 2D bounding boxes. Our 3D bed model was trained with two parts per face. Fig. 3 shows the statistics of the dataset in terms of the number of training examples for each aspect (where L-R-T denotes an aspect for which the front, right and the top face are visible), as well as per face. Note that the fact that the dataset is unbalanced (fewer examples for aspects with two faces) does not affect too much our approach, as only the face-stitching point deformation parameters are aspect specific. As we share the weights among the aspects, the number of training instances for each face is significantly higher (Fig. 3, middle). We compare this to DPM in Fig. 3, right. Our method can better exploit the training data by factoring out the viewpoint dependance of the training examples.

We begin our quantitative evaluation by using our model to reason about 2D detection. The 2D bounding boxes for our model are computed by fitting a 2D box around the convex hull of the projection of the predicted 3D box. We report average precision (AP) where we require that the output 2D boxes overlap with the ground-truth boxes at least $50\%$ using the intersection-over-union (IOU) criteria. The precision-recall curves are shown in Fig. 5. We compare our approach to the deformable part model (DPM) [1] and the cuboid model of Hedau et al. [2]. As shown in Table 1 we outperform the cuboid model of [2] by $8.1\%$ and DPM by $3.8\%$. This is notable, as to the best

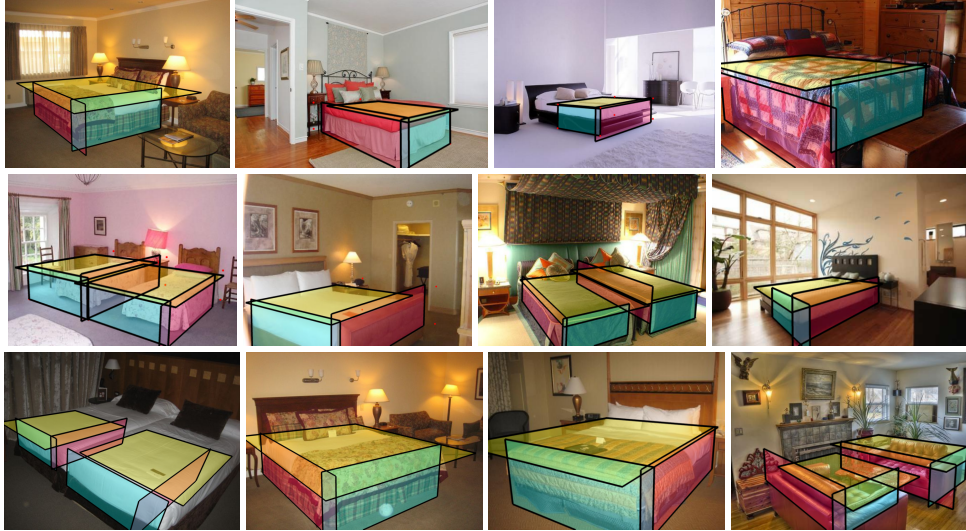

Figure 6: Detection examples obtained with our model on the bed dataset [2].

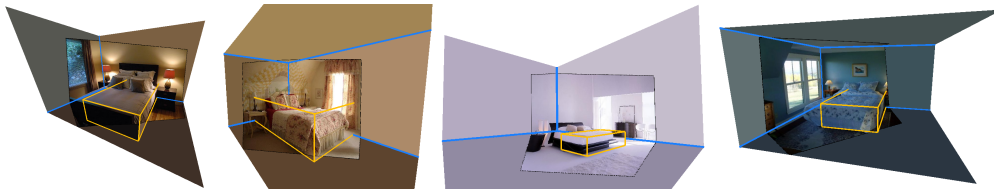

Figure 7: Detections in 3D + layout

of our knowledge, this is the first time that a 3D approach outperforms the DPM. [1] Examples of detections of our model are shown in Fig. 6.

A standard way to improve the detector's performance has been to rescore object detections using contextual information [1]. Following [2], we use two types of context. We first combined our detector with the 2D-DPM [1] to see whether the two sources of information complement each other. The second type of context is at the scene level, where we exploit the fact that the objects in indoor environments do not penetrate the walls and usually respect certain size ratios in 3D.

We combine the 3D and 2D detectors using a two step process, where first the 2D detector is run inside the bounding boxes produced by our cuboid model. A linear SVM that utilizes both scores as input is then employed to produce a score for the combined detection. While we observe a slight improvement in performance (1.1%), it seems that our cuboid model is already scoring the correct boxes well. This is in contrast to the cuboid model of [2], where the increase in performance is more significant due to the poorer accuracy of their 3D approach.

Following [2], we use an estimate of the room layout to rescore the object hypotheses at the scene level. We use the approach by Schwing et al. [28] to estimate the layout. To train the re-scoring classifier, we use the image-relative width and height features as in [1], footprint overlap between the 3D box and the floor as in [2] as well as 3D statistics such as distance between the object 3D box and the wall relative to the room height and the ratio between the object and room height in 3D. This further increases our performance by 5.2% (Table 1). Examples of 3D reconstruction of the room and our predicted 3D object hypotheses are shown in Fig. 7.

To evaluate the 3D performance of our detector we use the convex hull overlap measure as introduced in [2]. Here, instead of computing the overlap between the predicted boxes, we require that the convex hulls of our 3D hypotheses projected to the image plane and groundtruth annotations overlap at least 50% in IOU measure. Table 2 reports the results and shows that only little is lost in performance due to a stricter overlap measure.

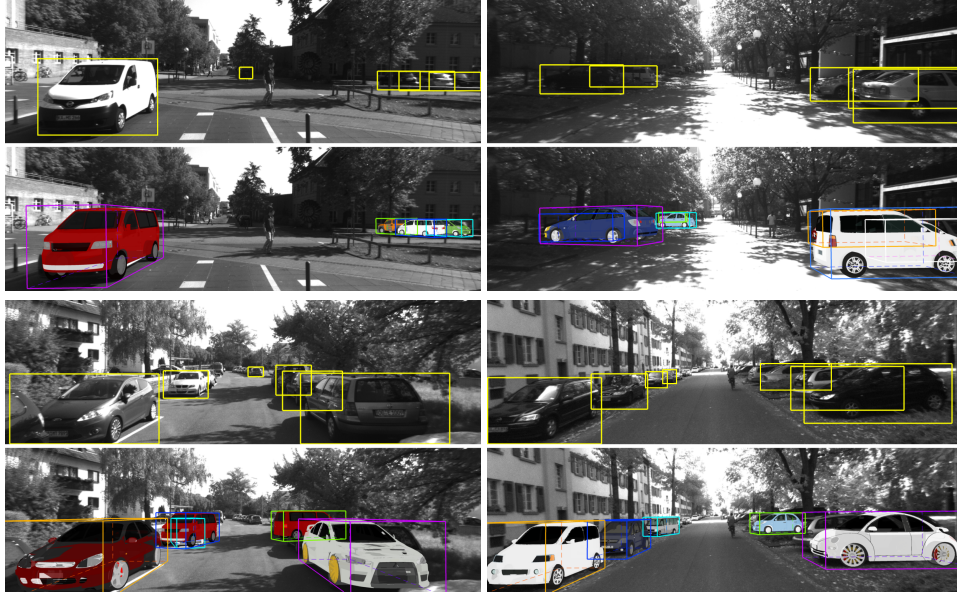

Figure 8: KITTI: examples of car detections. (**top**) Ground truth, (**bottom**) Our 3D detections, augmented with best fitting CAD models to visualize inferred 3D box orientations.

Since our model also predicts the locations of the dominant object faces (and thus the 3D object orientation), we would like to quantify its accuracy. We introduce an even stricter measure where we require that also the predicted cuboid faces overlap with the faces of the ground-truth cuboids. In particular, a hypothesis is correct if the average of the overlaps between top faces and vertical faces exceeds $50\%$ IOU. We compare the results of our approach to DPM [1]. Note however, that [1] returns only 2D boxes and hence a direct comparison is not possible. We thus augment the original DPM with 3D information in the following way. Since the three dominant orientations of the room, and thus the objects, are known (estimated via the vanishing points), we can find a 3D box whose projection best overlaps with the output of the 2D detector. This can be done by sliding a cuboid (whose dimensions match our cuboid model) in 3D to best fit the 2D bounding box. Our approach outperforms the 3D augmented DPM by a significant margin of $16.7\%$. We attribute this to the fact that our cuboid is deformable and thus the faces localize more accurately on the faces of the object.

We also conducted preliminary tests for our model on the autonomous driving dataset KITTI [16]. We trained our model with 8 aspects (estimated from the data) and 4 parts per face. An example of a learned aspect model is shown in Fig. 4. Note that the rectangular patches on the faces represent the parts, and color coding is used to depict the learned part and face deformation weights. We can observe that the model effectively and compactly factors out the appearance changes due to changes in viewpoint. Examples of detections are shown in Fig.8. The top rows show groundtruth annotations, while the bottom rows depict our predicted 3D boxes. To showcase also the viewpoint prediction of our detector we insert a CAD model inside each estimated 3D box, matching its orientation in 3D. In particular, for each detection we automatically chose a CAD model out of a collection of 80 models whose 3D bounding box best matches the dimensions of the predicted box. One can see that our 3D detector is able to predict the viewpoints of the objects well, as well as the type of car.

## 5 Conclusion

We proposed a novel approach to 3D object detection, which extends the well-acclaimed DPM to reason in 3D by means of a deformable 3D cuboid. Our cuboid allows for deformations at the face level via a stitching point as well as deformations between the faces and the parts. We demonstrated the effectiveness of our approach in indoor and outdoor scenarios and showed that our approach outperforms [1] and [2] in terms of 2D and 3D estimation. In future work, we plan to reason jointly about the 3D scene layout and the objects in order to improve the performance in both tasks.

**Acknowledgements.** S.F. has been supported in part by DARPA, contract number W911NF-10-2-0060. The views and conclusions contained in this document are those of the authors and should not be interpreted as representing the official policies, either express or implied, of the Army Research Laboratory or the U.S. Government.

## Footnotes

[1] Note that the numbers for our and [2]'s version of DPM slightly differ. The difference is likely due to how the negative examples are sampled during training (the dataset has a positive example in each training image).

# References

[1] Felzenszwalb, P. F., Girshick, R. B., McAllester, D., and Ramanan, D. (2010) Object detection with discriminatively trained part based models. *IEEE TPAMI*, **32**, 1627–1645.

[2] Hedau, V., Hoiem, D., and Forsyth, D. (2010) Thinking inside the box: Using appearance models and context based on room geometry. *ECCV*, vol. 6, pp. 224–237.

[3] Hinterstoisser, S., Lepetit, V., Ilic, S., Fua, P., and Navab, N. (2010) Dominant orientation templates for real-time detection of texture-less objects. *CVPR*.

[4] Schneiderman, H. and Kanade, T. (2000) A statistical method for 3d object detection applied to faces and cars. *CVPR*, pp. 1746–1759.

[5] Torralba, A., Murphy, K. P., and Freeman, W. T. (2007) Sharing visual features for multiclass and multi-view object detection. *IEEE TPAMI*, **29**, 854–869.

[6] Gu, C. and Ren, X. (2010) Discriminative mixture-of-templates for viewpoint classification. *ECCV*, pp. 408–421.

[7] Lowe, D. (1991) Fitting parameterized three-dimensional models to images. *IEEE TPAMI*, **13**, 441–450.

[8] Liebelt, J., Schmid, C., and Schertler, K. (2008) Viewpoint-independent object class detection using 3d feature maps. *CVPR*.

[9] Yan, P., Khan, S. M., and Shah, M. (2007) 3d model based oblect class detection in an arbitrary view. *ICCV*.

[10] Glasner, D., Galun, M., Alpert, S., Basri, R., and Shakhnarovich, G. (2011) Viewpoint-aware object detection and pose estimation. *ICCV*.

[11] Savarese, S. and Fei-Fei, L. (2007) 3d generic object categorization, localization and pose estimation. *ICCV*.

[12] Su, H., Sun, M., Fei-Fei, L., and Savarese, S. (2009) Learning a dense multi-view representation for detection, viewpoint classification and synthesis of object categories. *ICCV*.

[13] Pepik, B., Stark, M., Gehler, P., and Schiele, B. (2012) Teaching 3d geometry to deformable part models. Belongie, S., Blake, A., Luo, J., and Yuille, A. (eds.), *CVPR*.

[14] Dalal, N. and Triggs, B. (2005) Histograms of oriented gradients for human detection. *CVPR*.

[15] Koenderink, J. and van Doorn, A. (1976) The singularities of the visual mappings. *Bio. Cyber.*, **24**, 51–59.

[16] Geiger, A., Lenz, P., and Urtasun, R. (2012) Are we ready for autonomous driving? *CVPR*.

[17] Kushal, A., Schmid, C., and Ponce, J. (2007) Flexible object models for category-level 3d object recognition. *CVPR*.

[18] Thomas, A., Ferrari, V., Leibe, B., Tuytelaars, T., Schiele, B., and Gool, L. V. (2006) Toward multi-view object class detection. *CVPR*.

[19] Hoiem, D., Rother, C., and Winn, J. (2007) 3d layoutcrf for multi-view object class recognition and segmentation. *CVPR*.

[20] Sun, M., Su, H., Savarese, S., and Fei-Fei, L. (2009) A multi-view probabilistic model for 3d oblect classes. *CVPR*.

[21] Payet, N. and Todorovic, S. (2011) Probabilistic pose recovery using learned hierarchical object models. *ICCV*.

[22] Stark, M., Goesele, M., and Schiele, B. (2010) Back to the future: Learning shape models from 3d cad data. *British Machine Vision Conference*.

[23] Brooks, R. A. (1983) Model-based three-dimensional interpretations of two-dimensional images. *IEEE TPAMI*, **5**, 140–150.

[24] Dickinson, S. J., Pentland, A. P., and Rosenfeld, A. (1992) 3-d shape recovery using distributed aspect matching. *IEEE TPAMI*, **14**, 174–198.

[25] Sun, M., Bradski, G., Xu, B.-X., and Savarese, S. (2010) Depth-encoded hough voting for coherent object detection, pose estimation, and shape recovery. *ECCV*.

[26] Xiang, Y. and Savarese, S. (2012) Estimating the aspect layout of object categories. *CVPR*.

[27] Yu, C.-N. and Joachims, T. (2009) Learning structural svms with latent variables. *ICML*.

[28] Schwing, A., Hazan, T., Pollefeys, M., and Urtasun, R. (2012) Efficient structured prediction for 3d indoor scene understanding. *CVPR*.

